# Non-linear Prediction of Acoustic Vectors Using Hierarchical Mixtures of Experts

**S.R.Waterhouse**          **A.J.Robinson**

Cambridge University Engineering Department,
Trumpington St., Cambridge, CB2 1PZ, England.
Tel: [+44] 223 332800, Fax: [+44] 223 332662,
Email: srw1001, ajr @eng.cam.ac.uk
URL: http://svr-www.eng.cam.ac.uk/~srw1001

## Abstract

In this paper we consider speech coding as a problem of speech modelling. In particular, prediction of parameterised speech over short time segments is performed using the Hierarchical Mixture of Experts (HME) (Jordan & Jacobs 1994). The HME gives two advantages over traditional non-linear function approximators such as the Multi-Layer Perceptron (MLP); a statistical understanding of the operation of the predictor and provision of information about the performance of the predictor in the form of likelihood information and local error bars. These two issues are examined on both toy and real world problems of regression and time series prediction. In the speech coding context, we extend the principle of combining local predictions via the HME to a Vector Quantization scheme in which fixed local codebooks are combined on-line for each observation.

## 1 INTRODUCTION

We are concerned in this paper with the application of multiple models, specifically the Hierarchical Mixtures of Experts, to time series prediction, specifically the problem of predicting acoustic vectors for use in speech coding. There have been a number of applications of multiple models in time series prediction. A classic example is the *Threshold Autoregressive model* (TAR) which was used by Tong &

Lim (1980) to predict sunspot activity. More recently, Lewis, Kay and Stevens (*in* Weigend & Gershenfeld (1994)) describe the use of Multivariate and Regression Splines (MARS) to the prediction of future values of currency exchange rates. Finally, in speech prediction, Cuperman & Gersho (1985) describe the Switched Inter-frame Vector Prediction (SIVP) method which switches between separate linear predictors trained on different statistical classes of speech. The form of time series prediction we shall consider in this paper is the *single step prediction* $\hat{y}^{(t)}$ of a future quantity $y^{(t)}$, by considering the previous $p$ samples. This may be viewed as a regression problem over input-output pairs $\{x^{(t)}, y^{(t)}\}_{t=0}^{N}$ where $x^{(t)}$ is the *lag vector* $(y^{(t-1)}, y^{(t-2)}, ..., y^{(t-p)})$. We may perform this regression using standard linear models such as the Auto-Regressive (AR) model or via nonlinear models such as connectionist feed-forward or recurrent networks. The HME overcomes a number of problems associated with traditional connectionist models via its architecture and statistical framework. Recently, Jordan & Jacobs (1994) and Waterhouse & Robinson (1994) have shown that via the EM algorithm and a 2nd order optimization scheme known as Iteratively Reweighted Least Squares (IRLS), the HME is faster than standard Multilayer Perceptrons (MLP) by at least an order of magnitude on regression and classification tasks respectively. Jordan & Jacobs also describe various methods to visualise the learnt structure of the HME via 'deviance trees' and histograms of posterior probabilities. In this paper we provide further examples of the structural relationship of the trained HME and the input-output space in the form of *expert activation plots*. In addition we describe how the HME can be extended to give local error bars or measures of confidence in regression and time series prediction problems. Finally, we describe the extension of the HME to acoustic vector prediction, and a VQ coding scheme which utilises likelihood information from the HME.

## 2   HIERARCHICAL MIXTURES OF EXPERTS

The HME architecture (Figure 1) is based on the principle of 'divide and conquer' in which a large, hard to solve problem is broken up into many, smaller, easier to solve problems. It consists of a series of 'expert networks' which are trained on different parts of the input space. The outputs of the experts are combined by a 'gating network' which is trained to stochastically select the expert which is performing best at solving a particular part of the problem. The operation of the HME is as follows: the gating networks receive the input vectors $x^{(t)}$ and produce as outputs probabilities $P(m_j|x^{(t)}, \eta_j)$ for each local branch $m_j$ of assigning the current input to the different branches, where $\eta_j$ are the gating network parameters. The expert networks sit at the leaves of the tree and each output a vector $\hat{y}_j^{(t)}$ given input vector $x^{(t)}$ and parameters $\theta_j$. These outputs are combined in a weighted sum by $P(m_j|x^{(t)}, \eta_j)$ to give the overall output vector for this region. This procedure continues recursively upwards to the root node. In time series prediction, each expert $j$ is a linear single layer network with the form:

$$\hat{y}_j^{(t)} = \theta_j x^{(t)}$$

where $\theta_j$ is matrix and $x^{(t)}$ is the lag vector discussed earlier, which is identical in form to an AR model.

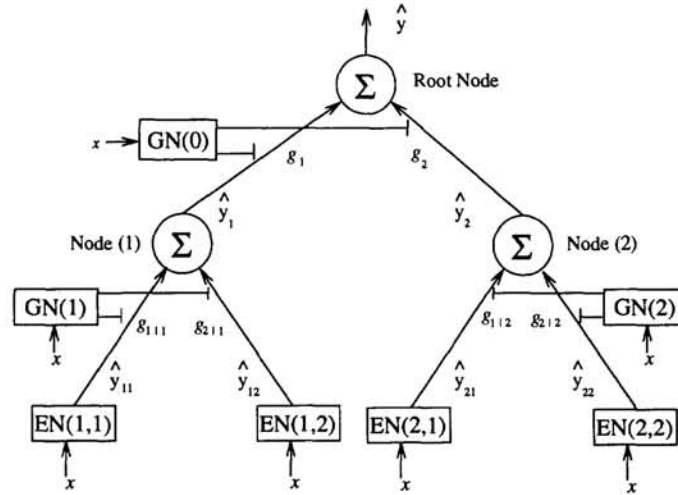

Figure 1: The Hierarchical Mixture of Experts.

## 2.1 Error bars via HME

Since each expert is an AR model, it follows that the output of each expert $\hat{y}^{(t)}$ is the expected value of the observations $y^{(t)}$ at each time $t$. The conditional likelihood of $y^{(t)}$ given the input and expert $m_j$ is

$$P(y^{(t)}|x^{(t)}, m_j, \theta_j) = \frac{1}{|2\pi C_j|} \exp\left(-\frac{1}{2}(y - \hat{y}_j^{(t)})^T C_j (y - \hat{y}_j^{(t)})\right)$$

where $C_j$ is the covariance matrix for expert $m_j$ which is updated during training as:

$$C_j = \frac{1}{\sum_t h_j^{(t)}} \sum_t h_j^{(t)}(y^{(t)} - \hat{y}_j^{(t)})^T (y^{(t)} - \hat{y}_j^{(t)})$$

where $h_j^{(t)}$ are the posterior probabilities[1] of each expert $m_j$. Taking the moments of the overall likelihood of the HME gives the output of the HME as the conditional expected value of the target output $y^{(t)}$,

$$
\begin{aligned}
\hat{y}^{(t)} &= E(y^{(t)}|x^{(t)}, \Theta, M) \\
&= \sum_j P(m_j|x^{(t)}, \eta_j) E(y^{(t)}|x^{(t)}, \theta_j, m_j) = \sum_j g_j^{(t)} \hat{y}_j^{(t)},
\end{aligned}
$$

Where $M$ represents the overall HME model and $\Theta$ the overall set of parameters. Taking the second central moment of $y^{(t)}$ gives,

$$
\begin{aligned}
C &= E((y^{(t)} - \hat{y}_j^{(t)})^2|x^{(t)}, \Theta, M) \\
&= \sum_j P(m_j|x^{(t)}, \eta_j) E((y^{(t)} - \hat{y}_j^{(t)})^2|x^{(t)}, \theta_j, m_j) \\
&= \sum_j g_j^{(t)}(C_j + \hat{y}_j^{(t)} \cdot \hat{y}_j^{(t)T}),
\end{aligned}
$$

which gives, in a direct fashion, the covariance of the output given the input and the model. If we assume that the observations are generated by an *underlying* model, which generates according to some function $f(x^{(t)})$ and corrupted by zero mean normally distributed noise $n(x)$ with constant covariance $\Sigma$, then the covariance of $y^{(t)}$ is given by,

$$V(y^{(t)}) = V(f^{(t)}) + \Sigma,$$

so that the covariance computed by the method above, $V(y^{(t)})$, takes into account the modelling error as well as the uncertainty due to the noise. Weigend & Nix (1994) also calculate error bars using an MLP consisting of a set of tanh hidden units to estimate the conditional mean and an auxiliary set of tanh hidden units to estimate the variance, assuming normally distributed errors. Our work differs in that there is no assumption of normality in the error distribution, rather that the errors of the terminal experts are distributed normally, with the total error distribution being a mixture of normal distributions.

## 3   SIMULATIONS

In order to demonstrate the utility of our approach to variance estimation we consider one toy regression problem and one time series prediction problem.

### 3.1   Toy Problem : Computer generated data

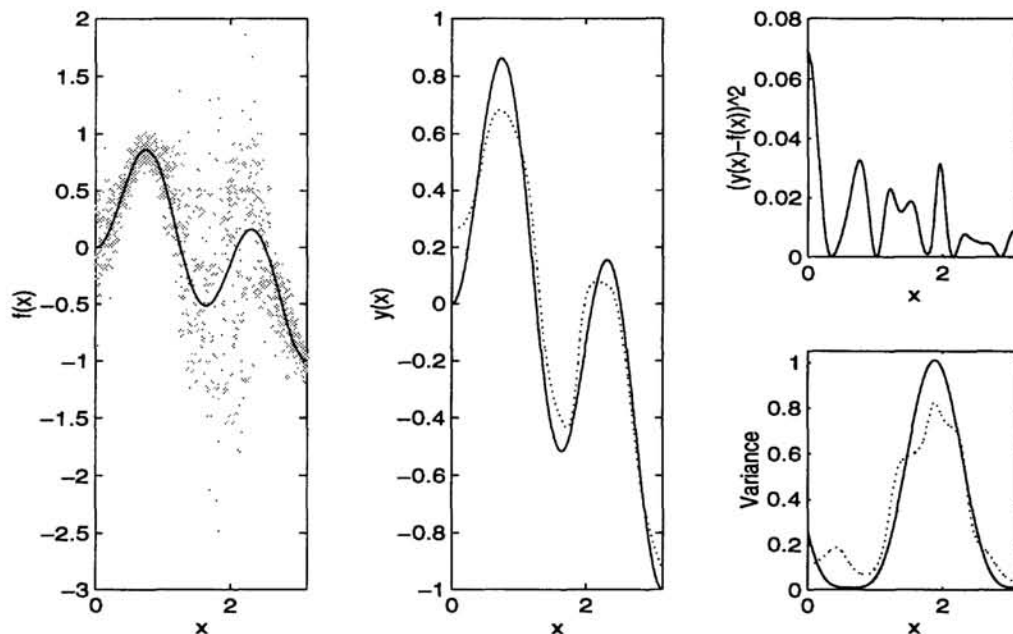

Figure 2: Performance on the toy data set of a 5 level binary HME. (a) training set (dots) and underlying function $f(x)$ (solid), (b) underlying function (solid) and prediction $y(x)$ (dashed), (c) squared deviation of prediction from underlying function, (d) true noise variance (solid) and variance of prediction (dashed).

By way of comparison, we used the same toy problem as Weigend & Nix (1994) which consists of 1000 training points and 10000 separate evaluation points from

the function $g(x)$ where $g(x)$ consists of a known underlying function $f(x)$ corrupted by normally distributed noise $N(0, \sigma^2(x))$,

$$f(x) = \sin(2.5x) \times \sin(1.5x), \quad \sigma^2(x) = 0.01 + 0.25 \times [1 - \sin(2.5x)]^2.$$

As can be seen by Figure 2, the HME has learnt to approximate both the underlying function and the additive noise variance. The deviation of the estimated variance from the "true" noise variance may be due to the actual noise variance being lower than the maximum denoted by the solid line at various points.

## 3.2 Sunspots

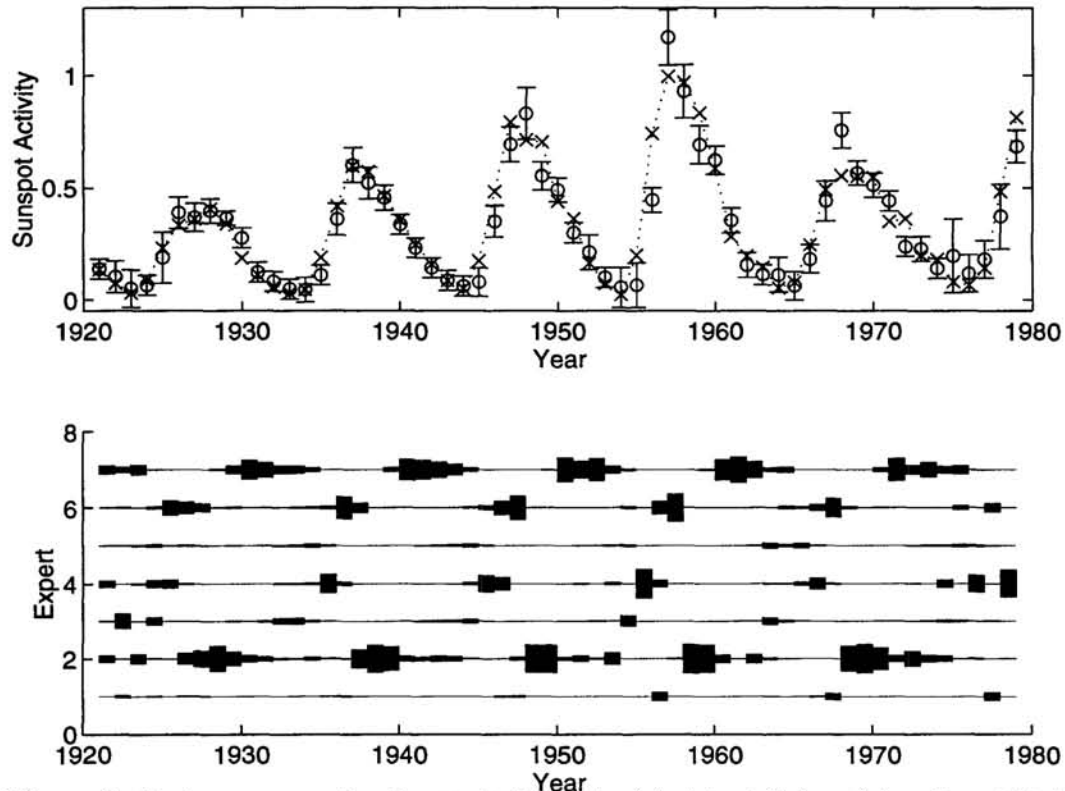

Figure 3: Performance on the **Sunspots** data set. (a) Actual Values (x) and predicted values (o) with error bars. (b) Activation of the expert networks; bars wide in the vertical axis indicate strong activation. Notice how expert 7 concentrates on the lulls in the series while expert 2 deals with the peaks.

| METHOD | NMSE' | | |
|---|---|---|---|
| | Train | Test | |
| | 1700-1920 | 1921-1955 | 1956-1979 |
| MLP | 0.082 | 0.086 | 0.35 |
| TAR | 0.097 | 0.097 | 0.28 |
| HME | 0.061 | 0.089 | 0.27 |

Table 1: Results of single step prediction on the **Sunspots** data set using a mixture of 7 experts (104 parameters) and a lag vector of 12 years. NMSE' is the NMSE normalised by the variance of the entire record 1700 to 1979.

The **Sunspots**[2] time series consists of yearly sunspot activity from 1700 to 1979 and was first tackled using connectionist models by Weigend, Huberman & Rumelhart (1990) who used a 12-8-1 MLP (113 parameters). Prior to this work, the TAR was used by Tong (1990). Our results, which were obtained using a random leave 10% out cross validation method, are shown in Table 1. We are considering only *single step* prediction on this problem, which involves prediction of the next value based on a set of previous values of the time series. Our results are evaluated in terms of *Normalised Mean Squared Error* (NMSE) (Weigend et al. 1990), which is defined as the ratio of the variance of the prediction on the test set to the variance of the test set itself.

The HME outperforms both the TAR and the MLP on this problem, and additionally provides both information about the structure of the network after training via the expert activation plot and error bars of the predictions, as shown in Figure 3. Further improvements may be possible by using likelihood information during cross validation so that a joint optimisation of overall error and variance is achieved.

## 4  SPEECH CODING USING HME

In the standard method of Linear Predictive Coding (LPC) (Makhoul 1975), speech is parametrised into a set of vectors of duration one frame (around 10 ms). Whilst simple scalar quantization of the LPC vectors can achieve bit rates of around 2400 bits per second (bps), Yong, Davidson & Gersho (1988) have shown that simple linear prediction of Line Spectral Pairs (LSP) (Soong & Juang 1984) vectors followed by Vector Quantization (VQ) (Abut, Gray & Rebolledo 1984) of the error vectors can yield bit rates of around 800 bps. In this paper we describe a speech coding framework which uses the HME in two stages. Firstly, the HME is used to perform prediction of the acoustic vectors. The error vectors are then quantized efficiently by using a VQ scheme which utilises the likelihood information derived from the HME.

### 4.1  Mixing VQ codebooks ia Gating networks

In a VQ scheme using a Euclidean distance measure, there is an implicit assumption that the inputs follow a Gaussian probability density function (pdf). This is satisfied if we quantize the residuals from a linear predictor, but not the residuals from an HME which follow a mixture of Gaussians pdf. A more efficient method is therefore to generate separate VQ codebooks for each expert in the HME and combine them via the priors on each expert from the gating networks. The codebook for the overall residual vectors on the test set is then generated at each time dynamically by choosing the first $D \times g_j^{(t)}$ codes, where $D$ is the size of the expert codebooks and $g_j^{(t)}$ is the prior on each expert.

## 4.2 Results of Speech Coding Evaluations

Initial experiments were performed using 23 Mel scale log energy frequency bins as acoustic vectors and using single variances $C_j = \sigma_j I$ as expert network covariance matrices. The results of training over 100,000 frames and evaluation over a further 100,000 frames on the Resource Management (RM) corpus are shown in Table 2 and Figure 4 which shows the good specialisation of the HME in this problem.

| METHOD | Prediction Gain (dB) | |
|---|---|---|
| | Train | Test |
| Linear | 12.07 | 10.95 |
| 1 level HME | 18.1 | 15.55 |
| 2 level HME | 20.20 | 16.39 |

Table 2: Prediction of Acoustic Vectors using linear prediction and binary branching HMEs with 1 and 2 levels. Prediction gain (Cuperman & Gersho 1985) is the ratio of the signal variance to prediction error variance.

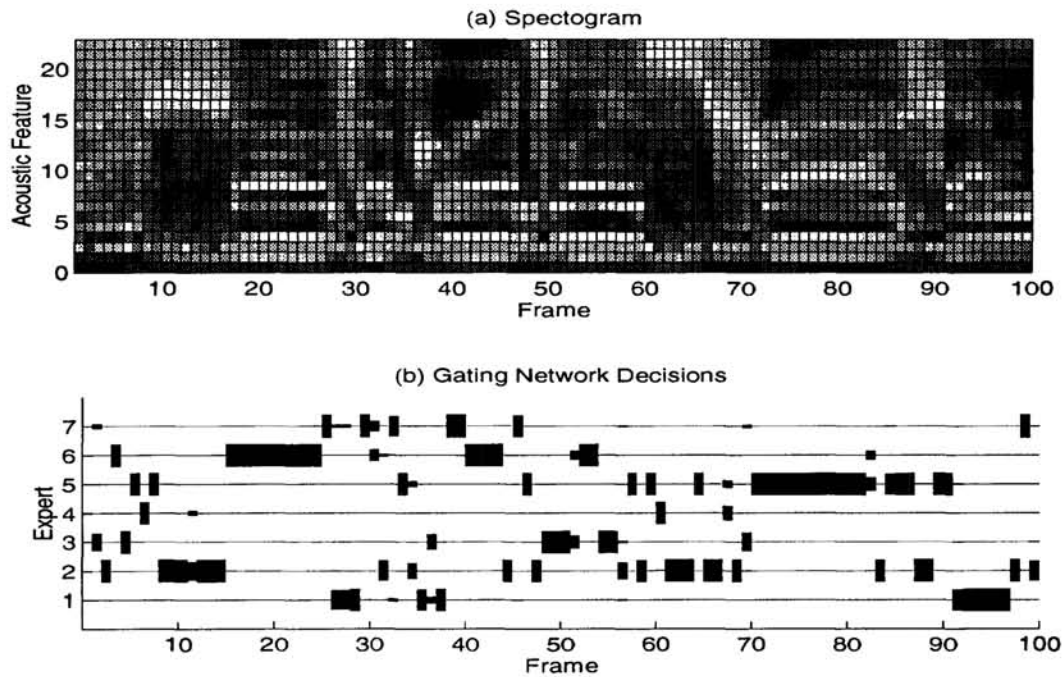

Figure 4: The behaviour of a mixture of 7 experts at predicting Mel-scale log energy frequency bins over 100 16ms frames. The top figure is a spectrogram of the speech and the lower figure is an expert activation plot, showing the gating network decisions.

We have conducted further experiments using LSPs and cepstrals as acoustic vectors, and using diagonal expert network covariance matrices, on a very large speech corpus. However, initial experiments show only a small improvement in gain over a single linear predictor and further investigation is underway. We have also coded acoustic vectors using 8 bits per frame with frame lengths of 12.5 ms, passing power, pitch and degree of voicing as side band information, without appreciable distortion over simple LPC coding. A full system will include prediction of all acoustic

parameters and we anticipate further reductions on this initial figure with future developments.

## 5   CONCLUSION

The aim of speech coding is the efficient coding of the speech signal with little perceptual loss. This paper has described the use of the HME for acoustic vector prediction. We have shown that the HME can provide improved performance over a linear predictor and in addition it provides a time varying variance for the prediction error. The decomposition of the linear prediction problem into a solution via a mixture of experts also allows us to construct a VQ codebook on the fly by mixing the codebooks of the various experts.

We expect that the direct computation of the time varying nature of the prediction accuracy will find many applications. Within the acoustic vector prediction problem we would like to exploit this information by exploring the continuum between the fixed bit rate coder described here and a variable bit rate coder that produces constant spectral distortion.

### Acknowledgements

This work was funded in part by Hewlett Packard Laboratories, UK. Steve Waterhouse is supported by an EPSRC Research Studentship and Tony Robinson was supported by a EPSRC Advanced Research Fellowship.

## Footnotes

[1]See (Jordan & Jacobs 1994) for a fuller discussion of posterior probabilities and likelihoods in the context of the HME.

[2]Available via anonymous ftp at **ftp.cs.colorado.edu** in /pub/Time-Series as **DataSunspots.Yearly**

## References

Abut, H., Gray, R. M. & Rebolledo, G. (1984), 'Vector quantization of speech and speech-like waveforms', *IEEE Transactions on Acoustics, Speech, and Signal Processing*.

Cuperman, V. & Gersho, A. (1985), 'Vector predictive coding of speech at 16 kbit/s', *IEEE Transactions on Communications* **COM-33**, 685–696.

Jordan, M. I. & Jacobs, R. A. (1994), 'Hierarchical Mixtures of Experts and the EM algorithm', *Neural Computation* **6**, 181–214.

Makhoul, J. (1975), 'Linear prediction: A tutorial review', *Proceedings of the IEEE* **63**(4), 561–580.

Soong, F. K. & Juang, B. H. (1984), Line spectrum pair (LSP) and speech data compression.

Tong, H. (1990), *Non-linear Time Series: a dynamical systems approach*, Oxford University Press.

Tong, H. & Lim, K. (1980), 'Threshold autoregression, limit cycles and cyclical data', *Journal of Royal Statistical Society*.

Waterhouse, S. R. & Robinson, A. J. (1994), Classification using hierarchical mixtures of experts, *in* 'IEEE Workshop on Neural Networks for Signal Processing'.

Weigend, A. S. & Gershenfeld, N. A. (1994), *Time Series Prediction: Forecasting the Future and Understanding the Past*, Addison-Wesley.

Weigend, A. S. & Nix, D. A. (1994), Predictions with confidence intervals (local error bars), Technical Report CU-CS-724-94, Department of Computer Science and Institute of Cognitive Science, University of Colorado, Boulder, CO 80309-0439.

Weigend, A. S., Huberman, B. A. & Rumelhart, D. E. (1990), 'Predicting the future: a connectionist approach', *International Journal of Neural Systems* **1**, 193–209.

Yong, M., Davidson, G. & Gersho, A. (1988), Encoding of LPC spectral parameters using switched-adaptive interframe vector prediction, *in* 'Proceedings of the IEEE International Conference on Acoustics Speech, and Signal Processing', pp. 402–405.